# Lower Bounds for Passive and Active Learning

**Maxim Raginsky**[*]
Coordinated Science Laboratory
University of Illinois at Urbana-Champaign

**Alexander Rakhlin**
Department of Statistics
University of Pennsylvania

## Abstract

We develop unified information-theoretic machinery for deriving lower bounds for passive and active learning schemes. Our bounds involve the so-called Alexander's capacity function. The supremum of this function has been recently rediscovered by Hanneke in the context of active learning under the name of "disagreement coefficient." For passive learning, our lower bounds match the upper bounds of Giné and Koltchinskii up to constants and generalize analogous results of Massart and Nédélec. For active learning, we provide first known lower bounds based on the capacity function rather than the disagreement coefficient.

## 1 Introduction

Not all Vapnik-Chervonenkis classes are created equal. This was observed by Massart and Nédélec [24], who showed that, when it comes to binary classification rates on a sample of size $n$ under a margin condition, some classes admit rates of the order $1/n$ while others only $(\log n)/n$. The latter classes were called "rich" in [24]. As noted by Giné and Koltchinskii [15], the fine complexity notion that defines this "richness" is in fact embodied in *Alexander's capacity function*.[1] Somewhat surprisingly, the supremum of this function (called *the disagreement coefficient* by Hanneke [19]) plays a key role in risk bounds for *active* learning. The contribution of this paper is twofold. First, we prove lower bounds for passive learning based on Alexander's capacity function, matching the upper bounds of [15] up to constants. Second, we prove lower bounds for the number of label requests in active learning in terms of the capacity function. Our proof techniques are information-theoretic in nature and provide a unified tool to study active and passive learning within the same framework.

**Active and passive learning.** Let $(\mathcal{X}, \mathcal{A})$ be an arbitrary measurable space. Let $(X, Y)$ be a random variable taking values in $\mathcal{X} \times \{0, 1\}$ according to an unknown distribution $P = \Pi \otimes P_{Y|X}$, where $\Pi$ denotes the marginal distribution of $X$. Here, $X$ is an *instance* (or a *feature*, a *predictor variable*) and $Y$ is a binary *response* (or a *label*). Classical results in statistical learning assume availability of an i.i.d. sample $\{(X_i, Y_i)\}_{i=1}^n$ from $P$. In this framework, the learner is *passive* and has no control on how this sample is chosen. The classical setting is well studied, and the following question has recently received attention: do we gain anything if data are obtained *sequentially*, and the learner is allowed to modify the *design distribution* $\Pi$ of the predictor variable before receiving the next pair $(X_i, Y_i)$? That is, can the learner *actively* use the information obtained so far to facilitate faster learning?

Two paradigms often appear in the literature: (i) the design distribution is a Dirac delta function at some $x_i$ that depends on $(x^{i-1}, Y^{i-1})$, or (ii) the design distribution is a restriction of the original distribution to some measurable set. There is rich literature on both approaches, and we only mention a few results here. The paradigm (i) is closely related to learning with membership queries [21], generalized binary search [25], and coding with noiseless feedback [6]. The goal is to actively choose the next $x_i$ so that the observed $Y_i \sim P_{Y|X=x_i}$ is sufficiently "informative" for the classification task. In this paradigm, the sample no longer provides information about the distribution

---

[*]Affiliation until January, 2012: Department of Electrical and Computer Engineering, Duke University.
[1]To be precise, the capacity function depends on the underlying probability distribution.

$\Pi$ (see [7] for further discussion and references). The setting (ii) is often called *selective sampling* [9, 13, 8], although the term *active learning* is also used. In this paradigm, the aim is to sequentially choose subsets $D_i \subseteq \mathcal{X}$ based on the observations prior to the $i$th example, such that the label $Y_i$ is requested only if $X_i \in D_i$. The sequence $\{X_i\}_{i=1}^n$ is assumed to be i.i.d., and so, form the view point of the learner, the $X_i$ is sampled from the conditional distribution $\Pi(\cdot|D_i)$.

In recent years, several interesting algorithms for active learning and selective sampling have appeared in the literature, most notably: the $A^2$ algorithm of Balcan et al. [4], which explicitly maintains $D_i$ as a "disagreement" set of a "version space"; the empirical risk minimization (ERM) based algorithm of Dasgupta et al. [11], which maintains the set $D_i$ implicitly through synthetic and real examples; and the importance-weighted active learning algorithm of Beygelzimer et al. [5], which constructs the design distribution through careful reweighting in the feature space. An insightful analysis has been carried out by Hanneke [20, 19], who distilled the role of the so-called *disagreement coefficient* in governing the performance of several of these active learning algorithms. Finally, Koltchinskii [23] analyzed active learning procedures using localized Rademacher complexities and Alexander's capacity function, which we discuss next.

**Alexander's capacity function.** Let $\mathcal{F}$ denote a class of candidate classifiers, where a *classifier* is a measurable function $f : \mathcal{X} \to \{0, 1\}$. Suppose the VC dimension of $\mathcal{F}$ is finite: VC-dim$(\mathcal{F}) = d$. The *loss* (or *risk*) of $f$ is its probability of error, $R_P(f) \triangleq \mathbb{E}_P[\mathbf{1}_{\{f(X) \neq Y\}}] = P(f(X) \neq Y)$. It is well known that the risk is globally minimized by the *Bayes classifier* $f^* = f_P^*$, defined by $f^*(x) \triangleq \mathbf{1}_{\{2\eta(x) \geq 1\}}$, where $\eta(x) \triangleq \mathbb{E}[Y|X = x]$ is the *regression function*. Define the *margin* as $h \triangleq \inf_{x \in \mathcal{X}} |2\eta(x) - 1|$. If $h > 0$, we say the problem satisfies *Massart's noise condition*. We define the *excess risk* of a classifier $f$ by $E_P(f) \triangleq R_P(f) - R_P(f^*)$, so that $E_P(f) \geq 0$, with equality if and only if $f = f^*$ $\Pi$-a.s. Given $\varepsilon \in (0, 1]$, define

$$\mathcal{F}_\varepsilon(f^*) \triangleq \{f \in \mathcal{F} : \Pi(f(X) \neq f^*(X)) \leq \varepsilon\},$$
$$D_\varepsilon(f^*) \triangleq \{x \in \mathcal{X} : \exists f \in \mathcal{F}_\varepsilon(f^*) \text{ s.t. } f(x) \neq f^*(x)\}$$

The set $\mathcal{F}_\varepsilon$ consists of all classifiers $f \in \mathcal{F}$ that are $\varepsilon$-close to $f^*$ in the $L_1(\Pi)$ sense, while the set $D_\varepsilon$ consists of all points $x \in \mathcal{X}$, for which there exists a classifier $f \in \mathcal{F}_\varepsilon$ that *disagrees* with the Bayes classifier $f^*$ at $x$. The *Alexander's capacity function* [15] is defined as

$$\tau(\varepsilon) \triangleq \Pi(D_\varepsilon(f^*))/\varepsilon, \tag{1}$$

that is, $\tau(\varepsilon)$ measures the relative size (in terms of $\Pi$) of the disagreement region $D_\varepsilon$ compared to $\varepsilon$. Clearly, $\tau(\varepsilon)$ is always bounded above by $1/\varepsilon$; however, in some cases $\tau(\varepsilon) \leq \tau_0$ with $\tau_0 < \infty$.

The function $\tau$ was originally introduced by Alexander [1, 2] in the context of exponential inequalities for empirical processes indexed by VC classes of functions, and Giné and Koltchinskii [15] generalized Alexander's results. In particular, they proved (see [15, p. 1213]) that, for a VC-class of binary-valued functions with VC-dim$(\mathcal{F}) = d$, the ERM solution $\widehat{f}_n = \arg\min_{f \in \mathcal{F}} \frac{1}{n} \sum_{i=1}^n \mathbf{1}_{\{f(X_i) \neq Y_i\}}$ under Massart's noise condition satisfies

$$E_P(\widehat{f}_n) \leq C \left[ \frac{d}{nh} \log \tau \left( \frac{d}{nh^2} \right) + \frac{s}{nh} \right] \tag{2}$$

with probability at least $1 - Ks^{-1}e^{-s/K}$ for some constants $C, K$ and any $s > 0$. The upper bound (2) suggests the importance of the Alexander's capacity function for passive learning, leaving open the question of necessity. Our first contribution is a lower bound which matches the upper bound (2) up to constant, showing that, in fact, dependence on the capacity is unavoidable.

Recently, Koltchinskii [23] made an important connection between Hanneke's disagreement coefficient and Alexander's capacity function. Under Massart's noise condition, Koltchinskii showed (see [23, Corollary 1]) that, for achieving an excess loss of $\varepsilon$ with confidence $1 - \delta$, the number of queries issued by his active learning algorithm is bounded above by

$$C \frac{\tau_0 \log(1/\varepsilon)}{h^2} \left[ d \log \tau_0 + \log(1/\delta) + \log \log(1/\varepsilon) + \log \log(1/h) \right], \tag{3}$$

where $\tau_0 = \sup_{\varepsilon \in (0,1]} \tau(\varepsilon)$ is Hanneke's disagreement coefficient. Similar bounds based on the disagreement coefficient have appeared in [19, 20, 11]. The second contribution of this paper is a *lower* bound on the expected number of queries based on Alexander's capacity $\tau(\varepsilon)$.

**Comparison to known lower bounds.** For passive learning, Massart and Nédélec [24] proved two lower bounds which, in fact, correspond to $\tau(\varepsilon) = 1/\varepsilon$ and $\tau(\varepsilon) = \tau_0$, the two endpoints on the complexity scale for the capacity function. Without the capacity function at hand, the authors emphasize that "rich" VC classes yield a larger lower bound. Our Theorem 1 below gives a unified construction for all possible complexities $\tau(\varepsilon)$.

In the PAC framework, the lower bound $\Omega(d/\varepsilon + (1/\varepsilon)\log(1/\delta))$ goes back to [12]. It follows from our results that in the noisy version of the problem ($h \neq 1$), the lower bound is in fact $\Omega((d/\varepsilon)\log(1/\varepsilon) + (1/\varepsilon)\log(1/\delta))$ for classes with $\tau(\varepsilon) = \Omega(1/\varepsilon)$.

For active learning, Castro and Nowak [7] derived lower bounds, but without the disagreement coefficient and under a Tsybakov-type noise condition. This setting is out of the scope of this paper. Hanneke [19] proved a lower bound on the number of label requests specifically for the $A^2$ algorithm in terms of the disagreement coefficient. In contrast, lower bounds of Theorem 2 are valid for *any* algorithm and are in terms of Alexander's capacity function. Finally, a result by Kääriäinen [22] (strengthened by [5]) gives a lower bound of $\Omega(\nu^2/\varepsilon^2)$ where $\nu = \inf_{f \in \mathcal{F}} E_P(f)$. A closer look at the construction of the lower bound reveals that it is achieved by considering a specific margin $h = \varepsilon/\nu$. Such an analysis is somewhat unsatisfying, as we would like to keep $h$ as a free parameter, not necessarily coupled with the desired accuracy $\varepsilon$. This point of view is put forth by Massart and Nédélec [24, p. 2329], who argue for a *non-asymptotic* analysis where all the parameters of the problem are made explicit. We also feel that this gives a better understanding of the problem.

## 2 Setup and main results

We suppose that the instance space $\mathcal{X}$ is a countably infinite set. Also, $\log(\cdot) \equiv \log_e(\cdot)$ throughout.

**Definition 1.** *Given a VC function class $\mathcal{F}$ and a margin parameter $h \in [0, 1]$, let $\mathfrak{C}(\mathcal{F}, h)$ denote the class of all conditional probability distributions $P_{Y|X}$ of $Y \in \{0, 1\}$ given $X \in \mathcal{X}$, such that: (a) the Bayes classifier $f^* \in \mathcal{F}$, and (b) the corresponding regression function satisfies the Massart condition with margin $h > 0$.*

Let $\mathcal{P}(\mathcal{X})$ denote the space of all probability measures on $\mathcal{X}$. We now introduce Alexander's capacity function (1) into the picture. Whenever we need to specify explicitly the dependence of $\tau(\varepsilon)$ on $f^*$ and $\Pi$, we will write $\tau(\varepsilon; f^*, \Pi)$. We also denote by $\mathcal{T}$ the set of all *admissible* capacity functions $\tau : (0, 1] \to \mathbb{R}^+$, i.e., $\tau \in \mathcal{T}$ if and only if there exist some $f^* \in \mathcal{F}$ and $\Pi \in \mathcal{P}(\mathcal{X})$, such that $\tau(\varepsilon) = \tau(\varepsilon; f^*, \Pi)$ for all $\varepsilon \in (0, 1]$. Without loss of generality, we assume $\tau(\varepsilon) \geq 2$.

**Definition 2.** *Given some $\Pi \in \mathcal{P}(\mathcal{X})$ and a pair $(\mathcal{F}, h)$ as in Def. 1, we let $\mathfrak{P}(\Pi, \mathcal{F}, h)$ denote the set of all joint distributions of $(X, Y) \in \mathcal{X} \times \{0, 1\}$ of the form $\Pi \otimes P_{Y|X}$, such that $P_{Y|X} \in \mathfrak{C}(\mathcal{F}, h)$. Moreover, given an admissible function $\tau \in \mathcal{T}$ and some $\varepsilon \in (0, 1]$, we let $\mathfrak{P}(\Pi, \mathcal{F}, h, \tau, \varepsilon)$ denote the subset of $\mathfrak{P}(\Pi, \mathcal{F}, h)$, such that $\tau(\varepsilon; f^*, \Pi) = \tau(\varepsilon)$.*

Finally, we specify the type of learning schemes we will be dealing with.

**Definition 3.** *An $n$-step learning scheme S consists of the following objects: $n$ conditional probability distributions $\Pi^{(t)}_{X_t|X^{t-1}, Y^{t-1}}$, $t = 1, \ldots, n$, and a mapping $\psi : \mathcal{X}^n \times \{0, 1\}^n \to \mathcal{F}$.*

This definition covers the passive case if we let

$$\Pi^{(t)}_{X_t|X^{t-1}, Y^{t-1}}(\cdot|x^{t-1}, y^{t-1}) = \Pi(\cdot), \qquad \forall(x^{t-1}, y^{t-1}) \in \mathcal{X}^{t-1} \times \{0, 1\}^{t-1}$$

as well as the active case, in which $\Pi^{(t)}_{X_t|X^{t-1}, Y^{t-1}}$ is the user-controlled design distribution for the feature at time $t$ given all currently available information. The learning process takes place sequentially as follows: At each time step $t = 1, \ldots, n$, a random feature $X_t$ is drawn according to $\Pi^{(t)}_{X^{t-1}, Y^{t-1}}(\cdot|X^{t-1}, Y^{t-1})$, and then a label $Y_t$ is drawn given $X_t$. After the $n$ samples $\{(X_t, Y_t)\}_{t=1}^n$ are collected, the learner computes the candidate classifier $\widehat{f}_n = \psi(X^n, Y^n)$.

To quantify the performance of such a scheme, we need the concept of an *induced measure*, which generalizes the set-up of [14]. Specifically, given some $P = \Pi \otimes P_{Y|X} \in \mathfrak{P}(\Pi, \mathcal{F}, h)$, define the

following probability measure on $\mathcal{X}^n \times \{0, 1\}^n$:

$$\mathbb{P}^{\mathsf{S}}(x^n, y^n) = \prod_{t=1}^{n} P_{Y|X}(y_t|x_t)\Pi_{X_t|X^{t-1},Y^{t-1}}^{(t)}(x_t|x^{t-1}, y^{t-1}).$$

**Definition 4.** *Let $\mathcal{Q}$ be a subset of $\mathfrak{P}(\Pi, \mathcal{F}, h)$. Given an accuracy parameter $\varepsilon \in (0, 1)$ and a confidence parameter $\delta \in (0, 1)$, an $n$-step learning scheme $\mathsf{S}$ is said to $(\varepsilon, \delta)$-learn $\mathcal{Q}$ if*

$$\sup_{P \in \mathcal{Q}} \mathbb{P}^{\mathsf{S}} \left( E_P(\widehat{f}_n) \geq \varepsilon h \right) \leq \delta. \tag{4}$$

**Remark 1.** Leaving the precision as $\varepsilon h$ makes the exposition a bit cleaner in light of the fact that, under Massart's noise condition with margin $h$, $E_P(f) \geq h\|f - f_P^*\|_{L_1(\Pi)} = h\Pi(f(X) \neq f_P^*(X))$ (cf. Massart and Nédélec [24, p. 2352]).

With these preliminaries out of the way, we can state the main results of this paper:

**Theorem 1** (Lower bounds for passive learning). *Given any $\tau \in \mathcal{T}$, any sufficiently large $d \in \mathbb{N}$ and any $\varepsilon \in (0, 1]$, there exist a probability measure $\Pi \in \mathcal{P}(\mathcal{X})$ and a VC class $\mathcal{F}$ with $\mathrm{VC\text{-}dim}(\mathcal{F}) = d$ with the following properties:*

*(1) Fix any $K > 1$ and $\delta \in (0, 1/2)$. If there exists an $n$-step passive learning scheme that $(\varepsilon/2, \delta)$-learns $\mathfrak{P}(\Pi, \mathcal{F}, h, \tau, \varepsilon)$ for some $h \in (0, 1 - K^{-1}]$, then*

$$n = \Omega\left(\frac{(1-\delta)d \log \tau(\varepsilon)}{K\varepsilon h^2} + \frac{\log \frac{1}{\delta}}{K\varepsilon h^2}\right). \tag{5}$$

*(2) If there exists an $n$-step passive learning scheme that $(\varepsilon/2, \delta)$-learns $\mathfrak{P}(\Pi, \mathcal{F}, 1, \tau, \varepsilon)$, then*

$$n = \Omega\left(\frac{(1-\delta)d}{\varepsilon}\right). \tag{6}$$

**Theorem 2** (Lower bounds for active learning). *Given any $\tau \in \mathcal{T}$, any sufficiently large $d \in \mathbb{N}$ and any $\varepsilon \in (0, 1]$, there exist a probability measure $\Pi \in \mathcal{P}(\mathcal{X})$ and a VC class $\mathcal{F}$ with $\mathrm{VC\text{-}dim}(\mathcal{F}) = d$ with the following property: Fix any $K > 1$ and any $\delta \in (0, 1/2)$. If there exists an $n$-step active learning scheme that $(\varepsilon/2, \delta)$-learns $\mathfrak{P}(\Pi, \mathcal{F}, h, \tau, \varepsilon)$ for some $h \in (0, 1 - K^{-1}]$, then*

$$n = \Omega\left(\frac{(1-\delta)d \log \tau(\varepsilon)}{Kh^2} + \frac{\tau(\varepsilon)\log \frac{1}{\delta}}{Kh^2}\right). \tag{7}$$

**Remark 2.** The lower bound in (6) is well-known and goes back to [12]. We mention it because it naturally arises from our construction. In fact, there is a smooth transition between (5) and (6), with the extra $\log \tau(\varepsilon)$ factor disappearing as $h$ approaches 1. As for the active learning lower bound, we conjecture that $d \log \tau(\varepsilon)$ is, in fact, optimal, and the extra factor of $\tau_0$ in $d\tau_0 \log \tau_0 \log(1/\varepsilon)$ in (3) arises from the use of a passive learning algorithm as a black box.

The remainder of the paper is organized as follows: Section 3 describes the required information-theoretic tools, which are then used in Section 4 to prove Theorems 1 and 2. The proofs of a number of technical lemmas can be found in the Supplementary Material.

## 3 Information-theoretic framework

Let $\mathbb{P}$ and $\mathbb{Q}$ be two probability distributions on a common measurable space $\mathcal{W}$. Given a convex function $\phi : [0, \infty) \to \mathbb{R}$ such that $\phi(1) = 0$, the *$\phi$-divergence*[2] between $\mathbb{P}$ and $\mathbb{Q}$ [3, 10] is given by

$$D_\phi(\mathbb{P}\|\mathbb{Q}) \triangleq \int_{\mathcal{W}} \frac{\mathrm{d}\mathbb{Q}}{\mathrm{d}\mu}\phi\left(\frac{\mathrm{d}\mathbb{P}/\mathrm{d}\mu}{\mathrm{d}\mathbb{Q}/\mathrm{d}\mu}\right)\mathrm{d}\mu, \tag{8}$$

where $\mu$ is an arbitrary $\sigma$-finite measure that dominates both $\mathbb{P}$ and $\mathbb{Q}$.[3] For the special case of $\mathcal{W} = \{0, 1\}$, when $\mathbb{P}$ and $\mathbb{Q}$ are the distributions of a Bernoulli$(p)$ and a Bernoulli$(q)$ random

variable, we will denote their $\phi$-divergence by

$$d_\phi(p\|q) = q \cdot \phi\left(\frac{p}{q}\right) + (1-q) \cdot \phi\left(\frac{1-p}{1-q}\right). \tag{9}$$

Two particular choices of $\phi$ are of interest: $\phi(u) = u \log u$, which gives the ordinary Kullback–Leibler (KL) divergence $D(\mathbb{P}\|\mathbb{Q})$, and $\phi(u) = -\log u$, which gives the *reverse* KL divergence $D(\mathbb{Q}\|\mathbb{P})$, which we will denote by $D_{\mathrm{re}}(\mathbb{P}\|\mathbb{Q})$. We will write $d(\cdot\|\cdot)$ for the binary KL divergence.

Our approach makes fundamental use of the *data processing inequality* that holds for any $\phi$-divergence [10]: if $\mathbb{P}$ and $\mathbb{Q}$ are two possible probability distributions for a random variable $W \in \mathcal{W}$ and if $P_{Z|W}$ is a conditional probability distribution of some other random variable $Z$ given $W$, then

$$D_\phi(\mathbb{P}_Z\|\mathbb{Q}_Z) \leq D_\phi(\mathbb{P}\|\mathbb{Q}), \tag{10}$$

where $\mathbb{P}_Z$ (resp., $\mathbb{Q}_Z$) is the marginal distribution of $Z$ when $W$ has distribution $\mathbb{P}$ (resp., $\mathbb{Q}$).

Consider now an arbitrary $n$-step learning scheme $\mathsf{S}$. Let us fix a finite set $\{f_1, \ldots, f_N\} \subset \mathcal{F}$ and assume that to each $m \in [N]$ we can associate a probability measure $P^m = \Pi \otimes P^m_{Y|X} \in \mathfrak{P}(\Pi, \mathcal{F}, h)$ with the Bayes classifier $f^*_{P_m} = f_m$. For each $m \in [N]$, let us define the induced measure

$$\mathbb{P}^{\mathsf{S},m}(x^n, y^n) \triangleq \prod_{t=1}^n P^m_{Y|X}(y_t|x_t)\Pi^{(t)}_{X_t|X^{t-1}, Y^{t-1}}(x_t|x^{t-1}, y^{t-1}). \tag{11}$$

Moreover, given any probability distribution $\pi$ over $[N]$, let $\mathbb{P}^{\mathsf{S},\pi}(m, x^n, y^n) \triangleq \pi(m)\mathbb{P}^{\mathsf{S},m}(x^n, y^n)$. In other words, $\mathbb{P}^{\mathsf{S},\pi}$ is the joint distribution of $(M, X^n, Y^n) \in [N] \times \mathcal{X}^n \times \{0, 1\}^n$, under which $M \sim \pi$ and $\mathbb{P}(X^n, Y^n|M = m) = \mathbb{P}^{\mathsf{S},m}(X^n, Y^n)$.

The first ingredient in our approach is standard [27, 14, 24]. Let $\{f_1, \ldots, f_N\}$ be an arbitrary $2\varepsilon$-packing subset of $\mathcal{F}$ (that is, $\|f_i - f_j\|_{L_1(\Pi)} > 2\varepsilon$ for all $i \neq j$). Suppose that $\mathsf{S}$ satisfies (4) on some $\mathcal{Q}$ that contains $\{P^1, \ldots, P^N\}$. Now consider

$$\widehat{M} \equiv \widehat{M}(X^n, Y^n) \triangleq \underset{1 \leq m \leq N}{\arg\min} \|\widehat{f}_n - f_m\|_{L_1(\Pi)}. \tag{12}$$

Then the following lemma is easily proved using triangle inequality:

**Lemma 1.** *With the above definitions, $\mathbb{P}^{\mathsf{S},\pi}(\widehat{M} \neq M) \leq \delta$.*

The second ingredient of our approach is an application of the data processing inequality (10) with a judicious choice of $\phi$. Let $W \triangleq (M, X^n, Y^n)$, let $M$ be uniformly distributed over $[N]$, $\pi(m) = \frac{1}{N}$ for all $m \in [N]$, and let $\mathbb{P}$ be the induced measure $\mathbb{P}^{\mathsf{S},\pi}$. Then we have the following lemma (see also [17, 16]):

**Lemma 2.** *Consider any probability measure $\mathbb{Q}$ for $W$, under which $M$ is distributed according to $\pi$ and independent of $(X^n, Y^n)$. Let the divergence-generating function $\phi$ be such that the mapping $p \mapsto d_\phi(p\|q)$ is nondecreasing on the interval $[q, 1]$. Then, assuming that $\delta \leq 1 - \frac{1}{N}$,*

$$D_\phi(\mathbb{P}\|\mathbb{Q}) \geq \frac{1}{N} \cdot \phi\left(N(1-\delta)\right) + \left(1 - \frac{1}{N}\right) \cdot \phi\left(\frac{N\delta}{N-1}\right). \tag{13}$$

*Proof.* Define the indicator random variable $Z = \mathbf{1}_{\{\widehat{M}=M\}}$. Then $\mathbb{P}(Z = 1) \geq 1 - \delta$ by Lemma 1. On the other hand, since $\mathbb{Q}$ can be factored as $\mathbb{Q}(m, x^n, y^n) = \frac{1}{N}\mathbb{Q}_{X^n,Y^n}(x^n, y^n)$, we have

$$\mathbb{Q}(Z = 1) = \sum_{m=1}^N \mathbb{Q}(M = m, \widehat{M} = m) = \frac{1}{N}\sum_{m=1}^N \sum_{x^n, y^n} \mathbb{Q}_{X^n, Y^n}(x^n, y^n)\mathbf{1}_{\{\widehat{M}(x^n, y^n)=m\}} = \frac{1}{N}.$$

Therefore,

$$D_\phi(\mathbb{P}\|\mathbb{Q}) \geq D_\phi(\mathbb{P}_Z\|\mathbb{Q}_Z) = d_\phi(\mathbb{P}(Z = 1)\|\mathbb{Q}(Z = 1)) \geq d_\phi(1 - \delta\|1/N),$$

where the first step is by the data processing inequality (10), the second is due to the fact that $Z$ is binary, and the third is by the assumed monotonicity property of $\phi$. Using (9), we arrive at (13). $\square$

Next, we need to choose the divergence-generating function $\phi$ and the auxiliary distribution $\mathbb{Q}$.

**Choice of $\phi$.** Inspection of the right-hand side of (13) suggests that the usual $\Omega(\log N)$ lower bounds [14, 27, 24] can be obtained if $\phi(u)$ behaves like $u \log u$ for large $u$. On the other hand, if $\phi(u)$ behaves like $-\log u$ for small $u$, then the lower bounds will be of the form $\Omega\left(\log \frac{1}{\delta}\right)$. These observations naturally lead to the respective choices $\phi(u) = u \log u$ and $\phi(u) = -\log u$, corresponding to the KL divergence $D(\mathbb{P}\|\mathbb{Q})$ and the reverse KL divergence $D_{\mathrm{re}}(\mathbb{P}\|\mathbb{Q}) = D(\mathbb{Q}\|\mathbb{P})$.

**Choice of $\mathbb{Q}$.** One obvious choice of $\mathbb{Q}$ satisfying the conditions of the lemma is the product of the marginals $\mathbb{P}_M \equiv \pi$ and $\mathbb{P}_{X^n,Y^n} \equiv N^{-1} \sum_{m=1}^{N} \mathbb{P}^{\mathsf{S},m}$: $\mathbb{Q} = \mathbb{P}_M \otimes \mathbb{P}_{X^n,Y^n}$. With this $\mathbb{Q}$ and $\phi(u) = u \log u$, the left-hand side of (13) is given by

$$D(\mathbb{P}\|\mathbb{Q}) = D(\mathbb{P}_{M,X^n,Y^n}\|\mathbb{P}_M \otimes \mathbb{P}_{X^n,Y^n}) = I(M; X^n, Y^n), \tag{14}$$

where $I(M; X^n, Y^n)$ is the mutual information between $M$ and $(X^n, Y^n)$ with joint distribution $\mathbb{P}$. On the other hand, it is not hard to show that the right-hand side of (13) can be lower-bounded by $(1 - \delta) \log N - \log 2$. Combining with (14), we get

$$I(M; X^n, Y^n) \geq (1 - \delta) \log N - \log 2,$$

which is (a commonly used variant of) the well-known Fano's inequality [14, Lemma 4.1], [18, p. 1250], [27, p. 1571]. The same steps, but with $\phi(u) = -\log u$, lead to the bound

$$L(M; X^n, Y^n) \geq \left(1 - \frac{1}{N}\right) \log \frac{1}{\delta} - \log 2 \geq \frac{1}{2} \log \frac{1}{\delta} - \log 2,$$

where $L(M; X^n, Y^n) \triangleq D_{\mathrm{re}}(\mathbb{P}_{M,X^n,Y^n}\|\mathbb{P}_M \otimes \mathbb{P}_{X^n,Y^n})$ is the so-called *lautum information* between $M$ and $(X^n, Y^n)$ [26], and the second inequality holds whenever $N \geq 2$.

However, it is often more convenient to choose $\mathbb{Q}$ as follows. Fix an arbitrary conditional distribution $Q_{Y|X}$ of $Y \in \{0, 1\}$ given $X \in \mathcal{X}$. Given a learning scheme $\mathsf{S}$, define the probability measure

$$\mathbb{Q}^{\mathsf{S}}(x^n, y^n) \triangleq \prod_{t=1}^{n} Q_{Y|X}(y_t|x_t)\Pi_{X_t|X^{t-1},Y^{t-1}}^{(t)}(x_t|x^{t-1}, y^{t-1}) \tag{15}$$

and let $\mathbb{Q}(m, x^n, y^n) = \frac{1}{N}\mathbb{Q}^{\mathsf{S}}(x^n, y^n)$ for all $m \in [N]$.

**Lemma 3.** *For each $x^n \in \mathcal{X}^n$ and $y \in \mathcal{X}$, let $N(y|x^n) \triangleq |\{1 \leq t \leq n : x_t = y\}|$. Then*

$$D(\mathbb{P}\|\mathbb{Q}) = \frac{1}{N} \sum_{m=1}^{N} \sum_{x \in \mathcal{X}} D(P_{Y|X}^m(\cdot|x)\|Q_{Y|X}(\cdot|x))\mathbb{E}_{\mathbb{P}^{\mathsf{S},m}}\left[N(x|X^n)\right]; \tag{16}$$

$$D_{\mathrm{re}}(\mathbb{P}\|\mathbb{Q}) = \frac{1}{N} \sum_{m=1}^{N} \sum_{x \in \mathcal{X}} D_{\mathrm{re}}(P_{Y|X}^m(\cdot|x)\|Q_{Y|X}(\cdot|x))\mathbb{E}_{\mathbb{Q}}\left[N(x|X^n)\right]. \tag{17}$$

*Moreover, if the scheme $\mathsf{S}$ is passive, then Eq. (17) becomes*

$$D_{\mathrm{re}}(\mathbb{P}\|\mathbb{Q}) = n \cdot \mathbb{E}_X \mathbb{E}_M\left[D_{\mathrm{re}}(P_{Y|X}^M(\cdot|X)\|Q_{Y|X}(\cdot|X))\right], \tag{18}$$

*and the same holds for $D_{\mathrm{re}}$ replaced by $D$.*

## 4 Proofs of Theorems 1 and 2

**Combinatorial preliminaries.** Given $k \in \mathbb{N}$, onsider the $k$-dimensional Boolean cube $\{0, 1\}^k = \{\beta = (\beta_1, \ldots, \beta_k) : \beta_i \in \{0, 1\}, i \in [k]\}$. For any two $\beta, \beta' \in \{0, 1\}^k$, define their Hamming distance $d_H(\beta, \beta') \triangleq \sum_{i=1}^{k} \mathbf{1}_{\{\beta_i \neq \beta_i'\}}$. The Hamming weight of any $\beta \in \{0, 1\}^k$ is the number of its nonzero coordinates. For $k > d$, let $\{0, 1\}_d^k$ denote the subset of $\{0, 1\}^k$ consisting of all binary strings with Hamming weight $d$. We are interested in large separated and well-balanced subsets of $\{0, 1\}_d^k$. To that end, we will use the following lemma:

**Lemma 4.** *Suppose that $d$ is even and $k > 2d$. Then, for $d$ sufficiently large, there exists a set $\mathcal{M}_{k,d} \subset \{0, 1\}_d^k$ with the following properties: (i) $\log |\mathcal{M}_{k,d}| \geq \frac{d}{4} \log \frac{k}{6d}$; (ii) $d_H(\beta, \beta') > d$ for any two distinct $\beta, \beta' \in \mathcal{M}_{k,d}^{(2)}$ ; (iii) for any $j \in [k]$,*

$$\frac{d}{2k} \leq \frac{1}{|\mathcal{M}_{k,d}|} \sum_{\beta \in \mathcal{M}_{k,d}} \beta_j \leq \frac{3d}{2k} \tag{19}$$

**Proof of Theorem 1.** Without loss of generality, we take $\mathcal{X} = \mathbb{N}$. Let $k = d\tau(\varepsilon)$ (we increase $\varepsilon$ if necessary to ensure that $k \in \mathbb{N}$), and consider the probability measure $\Pi$ that puts mass $\varepsilon/d$ on each $x = 1$ through $x = k$ and the remaining mass $1 - \varepsilon\tau(\varepsilon)$ on $x = k + 1$. (Recall that $\tau(\varepsilon) \leq 1/\varepsilon$.)

Let $\mathcal{F}$ be the class of indicator functions of all subsets of $\mathcal{X}$ with cardinality $d$. Then VC-dim$(\mathcal{F}) = d$. We will focus on a particular subclass $\mathcal{F}'$ of $\mathcal{F}$. For each $\beta \in \{0,1\}_d^k$, define $f_\beta : \mathcal{X} \to \{0,1\}$ by $f_\beta(x) = \beta_x$ if $x \in [k]$ and 0 otherwise, and take $\mathcal{F}' = \{f_\beta : \beta \in \{0,1\}_d^k\}$. For $p \in [0,1]$, let $\nu_p$ denote the probability distribution of a Bernoulli$(p)$ random variable. Now, to each $f_\beta \in \mathcal{F}'$ let us associate the following conditional probability measure $P_{Y|X}^\beta$:

$$P_{Y|X}^\beta(y|x) = \left[\nu_{(1+h)/2}(y)\beta_x + \nu_{(1-h)/2}(y)(1-\beta_x)\right]\mathbf{1}_{\{x\in[k]\}} + \mathbf{1}_{\{y=0\}}\mathbf{1}_{\{x\notin[k]\}}$$

It is easy to see that each $P_{Y|X}^\beta$ belongs to $\mathfrak{C}(\mathcal{F}, h)$. Moreover, for any two $f_\beta, f_{\beta'} \in \mathcal{F}$ we have

$$\|f_\beta - f_{\beta'}\|_{L_1(\Pi)} = \Pi(f_\beta(X) \neq f_{\beta'}(X)) = \frac{\varepsilon}{d}\sum_{i=1}^k \mathbf{1}_{\{\beta_i \neq \beta_i'\}} \equiv \frac{\varepsilon}{d}d_H(\beta, \beta').$$

Hence, for each choice of $f^* = f_{\beta^*} \in \mathcal{F}$ we have $\mathcal{F}_\varepsilon(f_{\beta^*}) = \{f_\beta : d_H(\beta, \beta^*) \leq d\}$. This implies that $D_\varepsilon(f_{\beta^*}) = [k]$, and therefore $\tau(\varepsilon; f_{\beta^*}, \Pi) = \Pi([k])/\varepsilon = \tau(\varepsilon)$. We have thus established that, for each $\beta \in \{0,1\}_d^k$, the probability measure $P^\beta = \Pi \otimes P_{Y|X}^\beta$ is an element of $\mathfrak{P}(\Pi, \mathcal{F}, h, \tau, \varepsilon)$. Finally, let $\mathcal{M}_{k,d} \subset \{0,1\}_d^k$ be the set described in Lemma 4, and let $\mathcal{G} \triangleq \{f_\beta : \beta \in \mathcal{M}_{k,d}\}$. Then for any two distinct $\beta, \beta' \in \mathcal{M}_{k,d}$ we have $\|f_\beta - f_{\beta'}\|_{L_1(\Pi)} = \frac{\varepsilon}{d}d_H(\beta, \beta') > \varepsilon$. Hence, $\mathcal{G}$ is a $\varepsilon$-packing of $\mathcal{F}'$ in the $L_1(\Pi)$-norm.

Now we are in a position to apply the lemmas of Section 3. Let $\{\beta^{(1)}, \ldots, \beta^{(N)}\}$, $N = |\mathcal{M}_{k,d}|$, be a fixed enumeration of the elements of $\mathcal{M}_{k,d}$. For each $m \in [N]$, let us denote by $P_{Y|X}^m$ the conditional probability measure $P_{Y|X}^{\beta^{(m)}}$, by $P^m$ the measure $\Pi \otimes P_{Y|X}^m$ on $\mathcal{X} \times \{0,1\}$, and by $f_m \in \mathcal{G}$ the corresponding Bayes classifier. Now consider any $n$-step passive learning scheme that $(\varepsilon/2, \delta)$-learns $\mathfrak{P}(\Pi, \mathcal{F}, h, \tau, \varepsilon)$, and define the probability measure $\mathbb{P}$ on $[N] \times \mathcal{X}^n \times \{0,1\}^n$ by $\mathbb{P}(m, x^n, y^n) = \frac{1}{N}\mathbb{P}^{\mathsf{S},m}(x^n, y^n)$, where $\mathbb{P}^{\mathsf{S},m}$ is constructed according to (11). In addition, for every $\gamma \in (0,1)$ define the auxiliary measure $\mathbb{Q}_\gamma$ on $[N] \times \mathcal{X}^n \times \{0,1\}^n$ by $\mathbb{Q}_\gamma(m, x^n, y^n) = \frac{1}{N}\mathbb{Q}_\gamma^{\mathsf{S}}(x^n, y^n)$, where $\mathbb{Q}_\gamma^{\mathsf{S}}$ is constructed according to (15) with

$$Q_{Y|X}^\gamma(y|x) \triangleq \nu_\gamma(y)\mathbf{1}_{\{x\in[k]\}} + \mathbf{1}_{\{y=0\}}\mathbf{1}_{\{x\notin[k]\}}.$$

Applying Lemma 2 with $\phi(u) = u\log u$, we can write

$$D(\mathbb{P}\|\mathbb{Q}_\gamma) \geq (1-\delta)\log N - \log 2 \geq \frac{(1-\delta)d}{4}\log\frac{k}{6d} - \log 2 \tag{20}$$

Next we apply Lemma 3. Defining $\eta = \frac{1+h}{2}$ and using the easily proved fact that

$$D(P_{Y|X}^m(\cdot|x)\|Q_{Y|X}^\gamma(\cdot|x)) = [d(\eta\|\gamma) - d(1-\eta\|\gamma)]f_m(x) + d(1-\eta\|\gamma)\mathbf{1}_{\{x\in[k]\}},$$

we get

$$D(\mathbb{P}\|\mathbb{Q}_\gamma) = n\varepsilon\left[d(\eta\|\gamma) + (\tau(\varepsilon)-1)d(1-\eta\|\gamma)\right]. \tag{21}$$

Therefore, combining Eqs. (20) and (21) and using the fact that $k = d\tau(\varepsilon)$, we obtain

$$n \geq \frac{(1-\delta)d\log\frac{\tau(\varepsilon)}{6} - \log 16}{4\varepsilon\left[d(\eta\|\gamma) + (\tau(\varepsilon)-1)d(1-\eta\|\gamma)\right]}, \qquad \forall\gamma \in (0,1) \tag{22}$$

This bound is valid for all $h \in (0,1]$, and the optimal choice of $\gamma$ for a given $h$ can be calculated in closed form: $\gamma^*(h) = \frac{1-h}{2} + \frac{h}{\tau(\varepsilon)}$. We now turn to the reverse KL divergence. First, suppose that $h \neq 1$. Lemma 2 gives $D_{\mathrm{re}}(\mathbb{P}\|\mathbb{Q}_{1-\eta}) \geq (1/2)\log(1/\delta) - \log 2$. On the other hand, using the fact that

$$D_{\mathrm{re}}(P_{Y|X}^m(\cdot|x)\|Q_{Y|X}^{1-\eta}(\cdot|x)) = d(\eta\|1-\eta)f_m(x) \tag{23}$$

and applying Eq. (18), we can write

$$D_{\text{re}}(\mathbb{P}\|\mathbb{Q}_{1-\eta}) = n\varepsilon \cdot d(\eta\|1-\eta) = n\varepsilon \cdot h \log \frac{1+h}{1-h}. \tag{24}$$

We conclude that

$$n \geq \frac{\frac{1}{2}\log\frac{1}{\delta} - \log 2}{\varepsilon h \log \frac{1+h}{1-h}}. \tag{25}$$

For $h = 1$, we get the vacuous bound $n \geq 0$.

Now we consider the two cases of Theorem 1.

(1) For a fixed $K > 1$, it follows from the inequality $\log u \leq u - 1$ that $h \log \frac{1+h}{1-h} \leq Kh^2$ for all $h \in (0, 1 - K^{-1}]$. Choosing $\gamma = \frac{1-h}{2}$ and using Eqs. (22) and (25), we obtain (5).

(2) For $h = 1$, we use (22) with the optimal setting $\gamma^*(1) = 1/\tau(\varepsilon)$, which gives (6). The transition between $h = 1$ and $h \neq 1$ is smooth and determined by $\gamma^*(h) = \frac{1-h}{2} + \frac{h}{\tau(\varepsilon)}$.

**Proof of Theorem 2.** We work with the same construction as in the proof of Theorem 1. First, let $\mathbb{Q}_{X^n,Y^n} \triangleq \frac{1}{N}\sum_{m=1}^{N}\mathbb{P}^{\mathsf{S},m}$. and $\mathbb{Q} = \pi \otimes \mathbb{Q}_{X^n,Y^n}$, where $\pi$ is the uniform distribution on $[N]$. Then, by convexity,

$$D(\mathbb{P}\|\mathbb{Q}) \leq \frac{1}{N^2}\sum_{m,m'=1}^{N}\mathbb{E}_{\mathbb{P}}\left[\sum_{t=1}^{n}\log\frac{P_{Y|X}^{m}(Y_t|X_t)}{P_{Y|X}^{m'}(Y_t|X_t)}\right] \leq n\max_{m,m'\in[N]}\max_{x\in[k]}D(P_{Y|X}^{m}(\cdot|x)\|P_{Y|X}^{m'}(\cdot|x))$$

which is upper bounded by $nh\log\frac{1+h}{1-h}$. Applying Lemma 2 with $\phi(u) = u \log u$, we therefore obtain

$$n \geq \frac{(1-\delta)d\log\frac{k}{6d} - \log 16}{4h\log\frac{1+h}{1-h}}. \tag{26}$$

Next, consider the auxiliary measure $\mathbb{Q}_{1-\eta}$ with $\eta = \frac{1+h}{2}$. Then

$$\begin{aligned}
D_{\text{re}}(\mathbb{P}\|\mathbb{Q}_{1-\eta}) &\overset{\text{(a)}}{=} \frac{1}{N}\sum_{M=1}^{N}\sum_{x=1}^{k}D_{\text{re}}(P_{Y|X}^{m}(\cdot|x)\|Q_{Y|X}^{1-\eta}(\cdot|x))\mathbb{E}_{\mathbb{Q}_{1-\eta}}[N(x|X^n)] \\
&\overset{\text{(b)}}{=} \frac{d(\eta\|1-\eta)}{N}\sum_{m=1}^{N}\sum_{x=1}^{k}f_m(x)\mathbb{E}_{\mathbb{Q}_{1-\eta}}[N(x|X^n)] \\
&= d(\eta\|1-\eta)\sum_{x=1}^{k}\left(\frac{1}{N}\sum_{m=1}^{N}f_m(x)\right)\mathbb{E}_{\mathbb{Q}_{1-\eta}}[N(x|X^n)] \\
&\overset{\text{(c)}}{=} d(\eta\|1-\eta)\sum_{x=1}^{k}\left(\frac{1}{N}\sum_{m=1}^{N}\beta_x^{(m)}\right)\mathbb{E}_{\mathbb{Q}_{1-\eta}}[N(x|X^n)] \\
&\overset{\text{(d)}}{\leq} \frac{3}{2\tau(\varepsilon)}h\log\frac{1+h}{1-h}\mathbb{E}_{\mathbb{Q}_{1-\eta}}\left[\sum_{x=1}^{k}N(x|X^n)\right] \\
&\overset{\text{(e)}}{\leq} \frac{3n}{2\tau(\varepsilon)}h\log\frac{1+h}{1-h},
\end{aligned}$$

where (a) is by Lemma 3, (b) is by (23), (c) is by definition of $\{f_m\}$, (d) is by the balance condition (19) satisfied by $\mathcal{M}_{k,d}$, and (e) is by the fact that $\sum_{x=1}^{k}N(x|X^n) \leq \sum_{x\in\mathcal{X}}N(x|X^n) = n$. Applying Lemma 2 with $\phi(u) = -\log u$, we get

$$n \geq \frac{\tau(\varepsilon)\left(\log\frac{1}{\delta} - \log 4\right)}{3h\log\frac{1+h}{1-h}} \tag{27}$$

Combining (26) and (27) and using the bound $h\log\frac{1+h}{1-h} \leq Kh^2$ for $h \in (0, 1 - K^{-1}]$, we get (7).

## Footnotes

[2]We deviate from the standard term "$f$-divergence" since $f$ is already reserved for a generic classifier.

[3]For instance, one can always take $\mu = \mathbb{P} + \mathbb{Q}$. It it easy to show that the value of $D_\phi(\mathbb{P}\|\mathbb{Q})$ in (8) does not depend on the choice of the dominating measure.

# References

[1] K.S. Alexander. Rates of growth and sample moduli for weighted empirical processes indexed by sets. *Probability Theory and Related Fields*, 75(3):379–423, 1987.

[2] K.S. Alexander. The central limit theorem for weighted empirical processes indexed by sets. *Journal of Multivariate Analysis*, 22(2):313–339, 1987.

[3] S. M. Ali and S. D. Silvey. A general class of coefficients of divergence of one distribution from another. *J. Roy. Stat. Soc. Ser. B*, 28:131–142, 1966.

[4] M.-F. Balcan, A. Beygelzimer, and J. Langford. Agnostic active learning. In *ICML '06: Proceedings of the 23rd international conference on Machine learning*, pages 65–72, New York, NY, USA, 2006. ACM.

[5] A. Beygelzimer, S. Dasgupta, and J. Langford. Importance weighted active learning. In *ICML*. ACM New York, NY, USA, 2009.

[6] M.V. Burnashev and K.S. Zigangirov. An interval estimation problem for controlled observations. *Problemy Peredachi Informatsii*, 10(3):51–61, 1974.

[7] R. M. Castro and R. D. Nowak. Minimax bounds for active learning. *IEEE Trans. Inform. Theory*, 54(5):2339–2353, 2008.

[8] G. Cavallanti, N. Cesa-Bianchi, and C. Gentile. Linear classification and selective sampling under low noise conditions. *Advances in Neural Information Processing Systems*, 21, 2009.

[9] D. Cohn, L. Atlas, and R. Ladner. Improving generalization with active learning. *Machine Learning*, 15(2):201–221, 1994.

[10] I. Csiszár. Information-type measures of difference of probability distributions and indirect observations. *Studia Sci. Math. Hungar.*, 2:299–318, 1967.

[11] S. Dasgupta, D. Hsu, and C. Monteleoni. A general agnostic active learning algorithm. In *Advances in neural information processing systems*, volume 20, page 2, 2007.

[12] A. Ehrenfeucht, D. Haussler, M. Kearns, and L. Valiant. A general lower bound on the number of examples needed for learning. *Information and Computation*, 82(3):247–261, 1989.

[13] Y. Freund, H.S. Seung, E. Shamir, and N. Tishby. Selective sampling using the query by committee algorithm. *Machine Learning*, 28(2):133–168, 1997.

[14] C. Gentile and D. P. Helmbold. Improved lower bounds for learning from noisy examples: an information-theoretic approach. *Inform. Comput.*, 166:133–155, 2001.

[15] E. Giné and V. Koltchinskii. Concentration inequalities and asymptotic results for ratio type empirical processes. *Ann. Statist.*, 34(3):1143–1216, 2006.

[16] A. Guntuboyina. Lower bounds for the minimax risk using $f$-divergences, and applications. *IEEE Trans. Inf. Theory*, 57(4):2386–2399, 2011.

[17] A. A. Gushchin. On Fano's lemma and similar inequalities for the minimax risk. *Theory of Probability and Mathematical Statistics*, pages 29–42, 2003.

[18] T. S. Han and S. Verdú. Generalizing the Fano inequality. *IEEE Trans. Inf. Theory*, 40(4):1247–1251, 1994.

[19] S. Hanneke. A bound on the label complexity of agnostic active learning. In *Proceedings of the 24th international conference on Machine learning*, page 360. ACM, 2007.

[20] S. Hanneke. Rates of convergence in active learning. *Ann. Statist.*, 39(1):333–361, 2011.

[21] T. Hegedűs. Generalized teaching dimensions and the query complexity of learning. In *COLT '95*, pages 108–117, New York, NY, USA, 1995. ACM.

[22] M. Kääriäinen. Active learning in the non-realizable case. In *ALT*, pages 63–77, 2006.

[23] V. Koltchinskii. Rademacher complexities and bounding the excess risk of active learning. *J. Machine Learn. Res.*, 11:2457–2485, 2010.

[24] P. Massart and É. Nédélec. Risk bounds for statistical learning. *Ann. Statist.*, 34(5):2326–2366, 2006.

[25] R. D. Nowak. The geometry of generalized binary search. Preprint, October 2009.

[26] D. P. Palomar and S. Verdú. Lautum information. *IEEE Trans. Inform. Theory*, 54(3):964–975, March 2008.

[27] Y. Yang and A. Barron. Information-theoretic determination of minimax rates of convergence. *Ann. Statist.*, 27(5):1564–1599, 1999.

